# NEWTRON: an Efficient Bandit algorithm for Online Multiclass Prediction

**Elad Hazan**
Department of Industrial Engineering
Technion - Israel Institute of Technology
Haifa 32000 Israel
ehazan@ie.technion.ac.il

**Satyen Kale**
Yahoo! Research
4301 Great America Parkway
Santa Clara, CA 95054
skale@yahoo-inc.com

## Abstract

We present an efficient algorithm for the problem of online multiclass prediction with bandit feedback in the fully adversarial setting. We measure its regret with respect to the log-loss defined in [AR09], which is parameterized by a scalar $\alpha$. We prove that the regret of NEWTRON is $O(\log T)$ when $\alpha$ is a constant that does not vary with horizon $T$, and at most $O(T^{2/3})$ if $\alpha$ is allowed to increase to infinity with $T$. For $\alpha = O(\log T)$, the regret is bounded by $O(\sqrt{T})$, thus solving the open problem of [KSST08, AR09]. Our algorithm is based on a novel application of the online Newton method [HAK07]. We test our algorithm and show it to perform well in experiments, even when $\alpha$ is a small constant.

## 1 Introduction

Classification is a fundamental task of machine learning, and is by now well understood in its basic variants. Unlike the well-studied supervised learning setting, in many recent applications (such as recommender systems, ad selection algorithms, etc.) we only obtain limited feedback about the true label of the input (e.g., in recommender systems, we only get feedback on the recommended items).

Several such problems can be cast as online, bandit versions of multiclass prediction problems[1]. The general framework, called the "contextual bandits" problem [LZ07], is as follows. In each round, the learner receives an input **x** in some high dimensional feature space (the "context"), and produces an action in response, and obtains an associated reward. The goal is to minimize regret with respect to a reference class of policies specifying actions for each context.

In this paper, we consider the special case of multiclass prediction, which is a fundamental problem in this area introduced by Kakade *et al* [KSST08]. Here, a learner obtains a feature vector, which is associated with an unknown label $y$ which can take one of $k$ values. Then the learner produces a prediction of the label, $\hat{y}$. In response, only 1 bit of information is given, whether the label is correct or incorrect. The goal is to design an efficient algorithm that minimizes regret with respect to a natural reference class of policies: linear predictors. Kakade *et al* [KSST08] gave an efficient algorithm, dubbed BANDITRON. Their algorithm attains regret of $O(T^{2/3})$ for a natural multiclass hinge loss, and they ask the question whether a better regret bound is possible. While the EXP4 algorithm [ACBFS03], applied to this setting, has an $O(\sqrt{T \log T})$ regret bound, it is highly inefficient, requiring $O(T^{n/2})$ time per iteration, where $n$ is the dimension of the feature space. Ideally, one would like to match or improve the $O(\sqrt{T \log T})$ regret bound of the EXP4 algorithm with an efficient algorithm (for a suitable loss function).

This question has received considerable attention. In COLT 2009, Abernethy and Rakhlin [AR09] formulated the open question precisely as minimizing regret for a suitable loss function in the fully

adversarial setting (and even offered a monetary reward for a resolution of the problem). Some special cases have been successfully resolved: the original paper of [KSST08], gives a $O(\sqrt{T})$ bound in the noiseless large-margin case. More recently, Crammer and Gentile [CG11] gave a $O(\sqrt{T} \log T)$ regret bound via an efficient algorithm based on the upper confidence bound method under a semi-adversarial assumption on the labels: they are generated stochastically via a specific linear model (with unknown parameters which change over time). Yet the general (fully adversarial) case has been unresolved as of now.

In this paper we address this question and design a novel algorithm for the fully adversarial setting with its expected regret measured with respect to log-loss function defined in [AR09], which is parameterized by a scalar $\alpha$. When $\alpha$ is a constant independent of $T$, we get a much stronger guarantee than required by the open problem: the regret is bounded by $O(\log T)$. In fact, the regret is bounded by $O(\sqrt{T})$ even for $\alpha = \Theta(\log T)$. Our regret bound for larger values of $\alpha$ increases smoothly to a maximum of $O(T^{2/3})$, matching that of BANDITRON in the worst case.

The algorithm is efficient to implement, and it is based on the online Newton method introduced in [HAK07]; hence we call the new algorithm NEWTRON. We implement the algorithm (and a faster variant, PNEWTRON) and test it on the same data sets used by Kakade *et al* [KSST08]. The experiments show improved performance over the BANDITRON algorithm, even for $\alpha$ as small as $10$.

# 2    Preliminaries

## 2.1    Notation

Let $[k]$ denote the set of integers $\{1, 2, \ldots, k\}$, and $\Delta_k \subseteq \mathbb{R}^k$ the set of distributions on $[k]$.

For any $\mathbb{R}^n$, let $\mathbf{1}, \mathbf{0}$ denote the all 1s and all 0s vectors respectively, and let $\mathbf{I}$ denote the identity matrix in $\mathbb{R}^{n \times n}$. For two (row or column) vectors $\mathbf{v}, \mathbf{w} \in \mathbb{R}^n$, we denote by $\mathbf{v} \cdot \mathbf{w}$ their usual inner product, i.e. $\mathbf{v} \cdot \mathbf{w} = \sum_{i=1}^{n} v_i w_i$. We denote by $\|\mathbf{v}\|$ the $\ell_2$ norm of $\mathbf{v}$. For a vector $\mathbf{v} \in \mathbb{R}^n$, denote by $\operatorname{diag}(\mathbf{v})$ the diagonal matrix in $\mathbb{R}^{n \times n}$ where the $i$th diagonal entry equals $v_i$.

For a matrix $\mathbf{W} \in \mathbb{R}^{k \times n}$, denote by $\mathbf{W}_1, \mathbf{W}_2, \ldots, \mathbf{W}_k$ its rows, which are (row) vectors in $\mathbb{R}^n$. To avoid defining unnecessary notation, we will interchangeably use $\mathbf{W}$ to denote both a matrix in $\mathbb{R}^{k \times n}$ or a (column) vector in $\mathbb{R}^{kn}$. The vector form of the matrix $\mathbf{W}$ is formed by arranging its rows one after the other, and then taking the transpose (i.e., the vector $[\mathbf{W}_1 | \mathbf{W}_2 | \cdots | \mathbf{W}_k]^\top$). Thus, for two matrices $\mathbf{V}$ and $\mathbf{W}$, $\mathbf{V} \cdot \mathbf{W}$ denotes their inner product in their vector form. For $i \in [n]$ and $l \in [k]$, denote by $\mathbf{E}_{il}$ the matrix which has 1 in its $(i, l)$th entry, and 0 everywhere else.

For a matrix $\mathbf{W}$, we denote by $\|\mathbf{W}\|$ the Frobenius norm of $\mathbf{W}$, which is also the usual $\ell_2$ norm of the vector form of $\mathbf{W}$, and so the notation is consistent. Also, we denote by $\|\mathbf{W}\|_2$ the spectral norm of $\mathbf{W}$, i.e. the largest singular value of $\mathbf{W}$.

For two matrices $\mathbf{W}$ and $\mathbf{V}$ denote by $\mathbf{W} \otimes \mathbf{V}$ their Kronecker product [HJ91]. For two square symmetric matrices $\mathbf{W}, \mathbf{V}$ of like order, denote by $\mathbf{W} \succeq \mathbf{V}$ the fact that $\mathbf{W} - \mathbf{V}$ is positive semidefinite, i.e. all its eigenvalues are non-negative. A useful fact of the Kronecker product is the following: if $\mathbf{W}, \mathbf{V}$ are symmetric matrices such that $\mathbf{W} \succeq \mathbf{V}$, and if $\mathbf{U}$ is a positive semidefinite symmetric matrix, then $\mathbf{W} \otimes \mathbf{U} \succeq \mathbf{V} \otimes \mathbf{U}$. This follows from the fact that if $\mathbf{W}, \mathbf{U}$ are both symmetric, positive semidefinite matrices, then so is their Kronecker product $\mathbf{W} \otimes \mathbf{U}$.

## 2.2    Problem setup

Learning proceeds in rounds. In each round $t$, for $t = 1, 2, \ldots, T$, we are presented a feature vector $\mathbf{x}_t \in \mathcal{X}$, where $\mathcal{X} \subseteq \mathbb{R}^n$, and $\|\mathbf{x}\| \leq R$ for all $\mathbf{x} \in \mathcal{X}$. Here $R$ is some specified constant. Associated with $\mathbf{x}_t$ is an unknown label $y_t \in [k]$. We are required to produce a prediction, $\hat{y}_t \in [k]$, as the label of $\mathbf{x}_t$. In response, we obtain only 1 bit of information: whether $\hat{y}_t = y_t$ or not. In particular, when $\hat{y}_t \neq y_t$, the identity of $y_t$ remains unknown (although one label, $\hat{y}_t$, is ruled out).

The learner's hypothesis class is parameterized by matrices $\mathbf{W} \in \mathbb{R}^{k \times n}$ with $\|\mathbf{W}\| \leq D$, for some specified constant $D$. Denote the set of such matrices by $\mathcal{K}$. Given a matrix $\mathbf{W} \in \mathcal{K}$ with the rows

$\mathbf{W}_1, \mathbf{W}_2, \ldots, \mathbf{W}_k$, the prediction associated with $\mathbf{W}$ for $\mathbf{x}_t$ is

$$\hat{y}_t = \arg\max_{i \in [k]} \mathbf{W}_i \cdot \mathbf{x}_t.$$

While ideally we would like to minimize the $0 - 1$ loss suffered by the learner, for computational reasons it is preferable to consider convex loss functions. A natural choice used in Kakade *et al* [KSST08] is the multi-class hinge loss:

$$\ell(\mathbf{W}, (\mathbf{x}_t, y_t)) = \max_{i \in [k] \setminus y_t} [1 - \mathbf{W}_{y_t} \cdot \mathbf{x}_t + \mathbf{W}_i \cdot \mathbf{x}_t]_+.$$

Other suitable loss functions $\ell(\cdot, \cdot)$ may also be used. The ultimate goal of the learner is to minimize regret, i.e.

$$\text{Regret} := \sum_{t=1}^{T} \ell(\mathbf{W}_t, (\mathbf{x}_t, y_t)) - \min_{\mathbf{W}^\star \in \mathcal{K}} \sum_{t=1}^{T} \ell(\mathbf{W}^\star, (\mathbf{x}_t, y_t)).$$

A different loss function was proposed in an open problem by Abernethy and Rakhlin in COLT 2009 [AR09]. We use this loss function in this paper and define it now.

We choose a constant $\alpha$ which parameterizes the loss function. Given a matrix $\mathbf{W} \in \mathcal{K}$ and an example $(\mathbf{x}, y) \in \mathcal{X} \times [k]$, define the function $\mathbf{P} : \mathcal{K} \times \mathcal{X} \to \Delta_k$ as

$$\mathbf{P}(\mathbf{W}, \mathbf{x})_i = \frac{\exp(\alpha \mathbf{W}_i \cdot \mathbf{x})}{\sum_j \exp(\alpha \mathbf{W}_j \cdot \mathbf{x})}.$$

Now let $\mathbf{p} = \mathbf{P}(\mathbf{W}, \mathbf{x})$. Suppose we make our prediction $\hat{y}_t$ by sampling from $\mathbf{p}$.

A natural loss function for this scheme is log-loss defined as follows:

$$\ell(\mathbf{W}, (\mathbf{x}, y)) = -\frac{1}{\alpha} \log(p_y) = -\frac{1}{\alpha} \log\left(\frac{\exp(\alpha \mathbf{W}_y \cdot \mathbf{x})}{\sum_j \exp(\alpha \mathbf{W}_k \cdot \mathbf{x})}\right)$$

$$= -\mathbf{W}_y \cdot \mathbf{x} + \frac{1}{\alpha} \log\left(\sum_j \exp(\alpha \mathbf{W}_j \cdot \mathbf{x})\right).$$

The log-loss is always positive. As $\alpha$ becomes large, this log-loss function has the property that when the prediction given by $\mathbf{W}$ for $\mathbf{x}$ is correct, it is very close to zero, and when the prediction is incorrect, it is roughly proportional to the margin of the incorrect prediction over the correct one.

The algorithm and its analysis depend upon the the gradient and Hessian of the loss function w.r.t. $\mathbf{W}$. The following lemma derives these quantities (proof in full version). Note that in the following, $\mathbf{W}$ is to be interpreted as a vector $\mathbf{W} \in \mathbb{R}^{kn}$.

**Lemma 1.** *Fix a matrix $\mathbf{W} \in \mathcal{K}$ and an example $(\mathbf{x}, y) \in \mathcal{X} \times [k]$, and let $\mathbf{p} = \mathbf{P}(\mathbf{W}, \mathbf{x})$. Then we have*

$$\nabla \ell(\mathbf{W}, (\mathbf{x}, y)) = (\mathbf{p} - \mathbf{e}_y) \otimes \mathbf{x} \; \text{ and } \; \nabla^2 \ell(\mathbf{W}, (\mathbf{x}, y)) = \alpha(\text{diag}(\mathbf{p}) - \mathbf{p}\mathbf{p}^\top) \otimes \mathbf{x}\mathbf{x}^\top.$$

In the analysis, we need bounds on the smallest non-zero eigenvalue of the $(\text{diag}(\mathbf{p}) - \mathbf{p}\mathbf{p}^\top)$ factor of the Hessian. Such bounds are given in the full version[2] For the sake of the analysis, however, the matrix inequality given in Lemma 2 below suffices. It is given in terms of a parameter $\delta$, which is the minimum probability of a label in any distribution $\mathbf{P}(\mathbf{W}, \mathbf{x})$.

**Definition 1.** *Define $\delta := \min_{\mathbf{W} \in \mathcal{K}, \mathbf{x} \in \mathcal{X}} \min_i \mathbf{P}(\mathbf{W}, \mathbf{x})_i$.*

We have the following (loose) bound on $\delta$, which follows easily using the fact that $|\mathbf{W}_i \cdot \mathbf{x}| \le RD$:

$$\delta \ge \exp(-2\alpha RD)/k. \tag{1}$$

**Lemma 2.** *Let $\mathbf{W} \in \mathcal{K}$ be any weight matrix, and let $\mathbf{H} \in \mathbb{R}^{k \times k}$ be any symmetric matrix such that $\mathbf{H}\mathbf{1} = \mathbf{0}$. Then we have*

$$\nabla^2 \ell(\mathbf{W}, (\mathbf{x}, y)) \succeq \frac{\alpha\delta}{\|\mathbf{H}\|_2} \mathbf{H} \otimes \mathbf{x}\mathbf{x}^\top.$$

**Algorithm 1** NEWTRON. Parameters: $\beta, \gamma$

1: Initialize $\mathbf{W}'_1 = \mathbf{0}$.
2: **for** $t = 1$ to $T$ **do**
3:     Obtain the example $\mathbf{x}_t$.
4:     Let $\mathbf{p}_t = \mathbf{P}(\mathbf{W}'_t, \mathbf{x}_t)$, and set $\mathbf{p}'_t = (1 - \gamma) \cdot \mathbf{p}_t + \frac{\gamma}{k}\mathbf{1}$.
5:     Output the label $\hat{y}_t$ by sampling from $\mathbf{p}'_t$. This is equivalent to playing $\mathbf{W}_t = \mathbf{W}'_t$ with probability $(1 - \gamma)$, and $\mathbf{W}_t = \mathbf{0}$ with probability $\gamma$.
6:     Obtain feedback, i.e. whether $\hat{y}_t = y_t$ or not.
7:     **if** $\hat{y}_t = y_t$ **then**
8:         Define $\tilde{\boldsymbol{\nabla}}_t := \frac{1-p_t(y_t)}{p'_t(y_t)} \cdot \left(\frac{1}{k}\mathbf{1} - \mathbf{e}_{y_t}\right) \otimes \mathbf{x}_t$ and $\kappa_t := p'_t(y_t)$.
9:     **else**
10:         Define $\tilde{\boldsymbol{\nabla}}_t := \frac{p_t(\hat{y}_t)}{p'_t(\hat{y}_t)} \cdot \left(\mathbf{e}_{\hat{y}_t} - \frac{1}{k}\mathbf{1}\right) \otimes \mathbf{x}_t$ and $\kappa_t := 1$.
11:     **end if**
12:     Define the cost function

$$f_t(\mathbf{W}) := \tilde{\boldsymbol{\nabla}}_t \cdot (\mathbf{W} - \mathbf{W}'_t) + \frac{1}{2}\kappa_t\beta(\tilde{\boldsymbol{\nabla}}_t \cdot (\mathbf{W} - \mathbf{W}'_t))^2. \tag{2}$$

13:     Compute

$$\mathbf{W}'_{t+1} := \arg\min_{\mathbf{W} \in \mathcal{K}} \sum_{\tau=1}^{t} f_t(\mathbf{W}) + \frac{1}{2D}\|\mathbf{W}\|^2. \tag{3}$$

14: **end for**

## 2.3 The FTAL Lemma

Our algorithm is based on the FTAL algorithm [HAK07]. This algorithm is an online version of the Newton step algorithm in offline optimization. The following lemma specifies the algorithm, specialized to our setting, and gives its regret bound. The proof is in the full version.

**Lemma 3.** *Consider an online convex optimization problem over some convex, compact domain $\mathcal{K} \subseteq \mathbb{R}^n$ of diameter $D$ with cost functions $f_t(\mathbf{w}) = (\mathbf{v}_t \cdot \mathbf{w} - \alpha_t) + \frac{1}{2}\beta_t(\mathbf{v}_t \cdot \mathbf{w} - \alpha_t)^2$, where the vector $\mathbf{v}_t \in \mathbb{R}^n$ and scalars $\alpha_t, \beta_t$ are chosen by the adversary such that for some known parameters $r, a, b$, we have $\|\mathbf{v}_t\| \leq r$, $\beta_t \geq a$, and $|\beta_t(\mathbf{v}_t \cdot \mathbf{w} - \alpha_t)| \leq b$, for all $\mathbf{w} \in \mathcal{K}$. Then the algorithm that, in round $t$, plays*

$$\mathbf{w}_t := \arg\min_{\mathbf{w} \in \mathcal{K}} \sum_{\tau=1}^{t-1} f_t(\mathbf{w})$$

*has regret bounded by $O(\frac{nb^2}{a} \log(\frac{DraT}{b}))$.*

## 3 The NEWTRON algorithm

Our algorithm for bandit multiclass learning algorithm, dubbed NEWTRON, is shown as Algorithm 1 above. In each iteration, we randomly choose a label from the distribution specified by the current weight matrix on the current example mixed with the uniform distribution over labels specified by an exploration parameter $\gamma$. The parameter $\gamma$ (which is similar to the exploration parameter used in the EXP3 algorithm of [ACBFS03]) is eventually tuned based on the value of the parameter $\alpha$ in the loss function (see Corollary 5). We then use the observed feedback to construct a quadratic loss function (which is strongly convex) that lower bounds the true loss function in expectation (see Lemma 7) and thus allows us to bound the regret. To do this we construct a randomized estimator $\tilde{\boldsymbol{\nabla}}_t$ for the gradient of the loss function at the current weight matrix. Furthermore, we also choose a parameter $\kappa_t$, which is an adjustment factor for the strongly convexity of the quadratic loss function ensuring that its expectation lower bounds the true loss function. Finally, we compute the new loss matrix using a Follow-The-Regularized-Leader strategy, by minimizing the sum of all quadratic loss functions so far with $\ell_2$ regularization. As described in [HAK07], this convex program can be solved in quadratic time, plus a projection on $\mathcal{K}$ in the norm induced by the Hessian.

**Statement and discussion of main theorem.** To simplify notation, define the function $\ell_t : \mathcal{K} \to \mathbb{R}$ as $\ell_t(\mathbf{W}) = \ell(\mathbf{W}, (\mathbf{x}_t, y_t))$. Let $\mathbb{E}_t[\cdot]$ denote the conditional expectation with respect to the $\sigma$-field $\mathcal{F}_t$, where $\mathcal{F}_t$ is the smallest $\sigma$-field with respect to which the predictions $\hat{y}_k$, for $k = 1, 2, \ldots, t-1$, are measurable.

With this notation, we can state our main theorem giving the regret bound:

**Theorem 4.** *Given $\alpha$, $\delta$ and $\gamma \leq \frac{1}{2}$, suppose we set $\beta \leq \min\{\frac{\alpha\delta}{10} + \eta, \frac{1}{4RD}\}$ in the* NEWTRON *algorithm, for $\eta = \frac{\gamma \log(k)}{20\alpha R^2 D^2}$. Let $\nu = \max\{\frac{\delta}{2}, \frac{\gamma}{k}\}$. The* NEWTRON *algorithm has the following bound on the expected regret:*

$$\sum_{t=1}^{T} \mathbb{E}[\ell_t(\mathbf{W}_t)] - \ell_t(\mathbf{W}^\star) = O\left(\frac{kn}{\nu\beta} \log T + \frac{\gamma \log(k)}{\alpha} T\right)$$

Before giving the proof theorem 4, we first state a corollary (a simple optimization of parameters, proved in the full version) which shows how $\gamma$ in Theorem 4 can be set appropriately to get a smooth interpolation between $O(\log(T))$ and $O(T^{2/3})$ regret based on the value of $\alpha$.

**Corollary 5.** *Given $\alpha$, there is a setting of $\gamma$ so that the regret of* NEWTRON *is bounded by*

$$\min\left\{c\frac{\exp(4\alpha RD)}{\alpha} \log(T), \quad 6cRDT^{2/3}\right\},$$

*where the constant $c = O(k^3 n)$ is independent of $\alpha$.*

**Discussion of the bound.** The parameter $\alpha$ is inherent to the log-loss function as defined in [AR09]. Our main result as given in Corollary 5 which entails logarithmic regret for constant $\alpha$, contains a constant which depends exponentially on $\alpha$. Empirically, it seems that $\alpha$ can be set to a small constant, say 10 (see Section 4), and still have good performance.

Note that even when $\alpha$ grows with $T$, as long as $\alpha \leq \frac{1}{8RD} \log(T)$, the regret can be bounded as $O(cRD\sqrt{T})$, thus solving the open problem of [KSST08, AR09] for log-loss functions with this range of $\alpha$.

We can say something even stronger - our results provide a "safety net" - no matter what the value of $\alpha$ is, the regret of our algorithm is never worse than $O(T^{2/3})$, matching the bound of the BANDITRON algorithm (although the latter holds for the multiclass hinge loss).

**Analysis.**

*Proof.* (Theorem 4.) The optimization (3) is essentially running the algorithm from Lemma 3 on $\mathcal{K}$ with the cost functions $f_t(\mathbf{W})$, with additional $nk$ initial fictitious cost functions $\frac{1}{2D}(\mathbf{E}_{il} \cdot \mathbf{W})^2$ for $i \in [n]$ and $l \in [k]$. These fictitious cost functions can be thought of as regularization. While technically these fictitious cost functions are not necessary to prove our regret bound, we include them since this seems to give better experimental performance and only adds a constant to the regret.

We now apply the regret bound of Lemma 3 by estimating the parameters $r, a, b$. This is a simple technical calculation and done in Lemma 6 below, which yields the values $r = \frac{R}{\nu}$, $a = \beta\nu$, $b = 1$. Hence, the regret bound of Lemma 3 implies that for any $\mathbf{W}^\star \in \mathcal{K}$,

$$\sum_{t=1}^{T} f_t(\mathbf{W}_t') - f_t(\mathbf{W}^\star) = O\left(\frac{kn}{\nu\beta} \log T\right).$$

Note that the bound above excludes the fictitious cost functions since they only add a constant additive term to the regret, which is absorbed by the $O(\log T)$ term. Similarly, we have also suppressed additive constants arising from the $\log(\frac{DraT}{b})$ term in the regret bound of Lemma 3.

Taking expectation on both sides of the above bound with respect to the randomness in the algorithm, and using the specification (2) of $f_t(\mathbf{W})$ we get

$$\mathbb{E}\left[\tilde{\boldsymbol{\nabla}}_t \cdot (\mathbf{W}_t' - \mathbf{W}^\star) - \frac{1}{2}\kappa_t\beta(\tilde{\boldsymbol{\nabla}}_t \cdot (\mathbf{W}_t' - \mathbf{W}^\star))^2\right] = O\left(\frac{kn}{\nu\beta} \log T\right). \qquad (4)$$

By Lemma 7 below, we get that

$$\ell_t(\mathbf{W}'_t) - \ell_t(\mathbf{W}^\star) \;\leq\; \mathbb{E}_t\left[\tilde{\boldsymbol{\nabla}}_t \cdot (\mathbf{W}'_t - \mathbf{W}^\star) - \frac{1}{2}\kappa_t\beta(\tilde{\boldsymbol{\nabla}}_t \cdot (\mathbf{W}'_t - \mathbf{W}^\star))^2\right] + 20\eta R^2 D^2. \quad (5)$$

Furthermore, we have

$$\mathbb{E}_t[\ell_t(\mathbf{W}_t)] - \ell(\mathbf{W}'_t) \;\leq\; \frac{\gamma \log(k)}{\alpha}, \quad (6)$$

since $\mathbf{W}_t = \mathbf{W}'_t$ with probability $(1 - \gamma)$ and $\mathbf{W}_t = \mathbf{0}$ with probability $\gamma$, and $\ell_t(\mathbf{0}) = \frac{\log(k)}{\alpha}$. Plugging (5) and (6) in (4), and using $\eta = \frac{\gamma \log(k)}{20\alpha R^2 D^2}$,

$$\sum_{t=1}^{T} \mathbb{E}[\ell(\mathbf{W}_t)] - \ell(\mathbf{W}^\star) \;=\; O\left(\frac{kn}{\nu\beta} \log T + \frac{\gamma \log(k)}{\alpha} T\right).$$

$\square$

We now state two lemmas that were used in the proof of Theorem 4. The first one (proof in the full version) obtains parameter settings to use Lemma 3 in Theorem 4.

**Lemma 6.** *Assume $\beta \leq \frac{1}{4RD}$ and $\gamma \leq \frac{1}{2}$. Let $\nu = \max\{\frac{\delta}{2}, \frac{\gamma}{k}\}$. Then the following are valid settings for the parameters $r, a, b$: $r = \frac{R}{\nu}$, $a = \beta\nu$ and $b = 1$.*

The next lemma shows that in each round, the expected regret of the inner FTAL algorithm with $f_t$ cost functions is larger than the regret of NEWTRON.

**Lemma 7.** *For $\beta = \frac{\alpha\delta}{10} + \eta$ and $\gamma \leq \frac{1}{2}$, we have*

$$\ell_t(\mathbf{W}'_t) - \ell_t(\mathbf{W}^\star) \;\leq\; \mathbb{E}_t\left[\tilde{\boldsymbol{\nabla}}_t \cdot (\mathbf{W}'_t - \mathbf{W}^\star) - \frac{1}{2}\kappa_t\beta(\tilde{\boldsymbol{\nabla}}_t \cdot (\mathbf{W}'_t - \mathbf{W}^\star))^2\right] + 20\eta R^2 D^2.$$

*Proof.* The intuition behind the proof is the following. We show that $\mathbb{E}_t[\tilde{\boldsymbol{\nabla}}_t] = (\mathbf{p} - \mathbf{e}_{y_t}) \otimes \mathbf{x}_t$, which by Lemma 1 equals $\nabla\ell_t(\mathbf{W}'_t)$. Next, we show that $\mathbb{E}_t[\kappa_t \tilde{\boldsymbol{\nabla}}_t \tilde{\boldsymbol{\nabla}}_t^\top] = \mathbf{H}_t \otimes \mathbf{x}_t \mathbf{x}_t^\top$ for some matrix $\mathbf{H}_t$ s.t. $\mathbf{H}_t \mathbf{1} = \mathbf{0}$. By upper bounding $\|\mathbf{H}_t\|$, we then show (using Lemma 2) that for any $\mathbf{\Psi} \in \mathcal{K}$ we have

$$\nabla^2\ell_t(\mathbf{\Psi}) \;\succeq\; \beta\mathbf{H}_t \otimes \mathbf{x}_t\mathbf{x}_t^\top.$$

The stated bound then follows by an application of Taylor's theorem.

The technical details for the proof are as follows. First, note that

$$\mathbb{E}_t[\tilde{\boldsymbol{\nabla}}_t \cdot (\mathbf{W}'_t - \mathbf{W}^\star)] = \mathbb{E}_t[\tilde{\boldsymbol{\nabla}}_t] \cdot (\mathbf{W}'_t - \mathbf{W}^\star). \quad (7)$$

We now compute $\mathbb{E}_t[\tilde{\boldsymbol{\nabla}}_t]$.

$$\mathbb{E}_t[\tilde{\boldsymbol{\nabla}}_t] = \left[p'_t(y_t) \cdot \frac{1 - p_t(y_t)}{p'_t(y_t)} \cdot \left(\frac{1}{k}\mathbf{1} - \mathbf{e}_{y_t}\right) + \sum_{y \neq y_t} p'_t(y) \cdot \frac{p_t(y)}{p'_t(y)} \cdot \left(\mathbf{e}_{\hat{y}_t} - \frac{1}{k}\mathbf{1}\right)\right] \otimes \mathbf{x}_t$$

$$= (\mathbf{p}_t - \mathbf{e}_{y_t}) \otimes \mathbf{x}_t. \quad (8)$$

Next, we have

$$\mathbb{E}_t[\kappa_t(\tilde{\boldsymbol{\nabla}}_t \cdot (\mathbf{W}'_t - \mathbf{W}^\star))^2] = (\mathbf{W}'_t - \mathbf{W}^\star)^\top \mathbb{E}_t[\kappa_t \tilde{\boldsymbol{\nabla}}_t \tilde{\boldsymbol{\nabla}}_t^\top](\mathbf{W}'_t - \mathbf{W}^\star). \quad (9)$$

We now compute $\mathbb{E}_t[\kappa_t \tilde{\boldsymbol{\nabla}}_t \tilde{\boldsymbol{\nabla}}_t^\top]$.

$$\mathbb{E}_t[\kappa_t \tilde{\boldsymbol{\nabla}}_t \tilde{\boldsymbol{\nabla}}_t^\top] = \left[p'_t(y_t) \cdot \kappa_t \left(\frac{1 - p_t(y_t)}{p'_t(y_t)}\right)^2 \cdot \left(\frac{1}{k}\mathbf{1} - \mathbf{e}_{y_t}\right)\left(\frac{1}{k}\mathbf{1} - \mathbf{e}_{y_t}\right)^\top\right.$$

$$\left. + \sum_{y \neq y_t} p'_t(y) \cdot \left(\frac{p_t(y)}{p'_t(y)}\right)^2 \cdot \left(\mathbf{e}_y - \frac{1}{k}\mathbf{1}\right)\left(\mathbf{e}_y - \frac{1}{k}\mathbf{1}\right)^\top\right] \otimes \mathbf{x}_t\mathbf{x}_t^\top$$

$$=: \mathbf{H}_t \otimes \mathbf{x}_t\mathbf{x}_t^\top, \quad (10)$$

where $\mathbf{H}_t$ is the matrix in the brackets above. We note a few facts about $\mathbf{H}_t$. First, note that $(\mathbf{e}_y - \frac{1}{k}\mathbf{1}) \cdot \mathbf{1} = 0$, and so $\mathbf{H}_t \mathbf{1} = 0$. Next, the spectral norm (i.e. largest eigenvalue) of $\mathbf{H}_t$ is bounded as:

$$\|\mathbf{H}_t\|_2 \leq \left\|\tfrac{1}{k}\mathbf{1} - \mathbf{e}_{y_t}\right\|^2 + \sum_{y \neq y_t} p_t'(y)\frac{1}{(1-\gamma)^2}\left\|\mathbf{e}_y - \tfrac{1}{k}\mathbf{1}\right\|^2 \leq 10,$$

for $\gamma \leq \frac{1}{2}$. Now, for any $\boldsymbol{\Psi} \in \mathcal{K}$, by Lemma 2, for the specified value of $\beta$ we have

$$\nabla^2 \ell_t(\boldsymbol{\Psi}) \succeq \frac{\alpha\delta}{10}\mathbf{H}_t \otimes \mathbf{x}_t \mathbf{x}_t^\top. \tag{11}$$

Now, by Taylor's theorem, for some $\boldsymbol{\Psi}$ on the line segment connecting $\mathbf{W}_t'$ to $\mathbf{W}^\star$, we have

$$\ell_t(\mathbf{W}^\star) - \ell_t(\mathbf{W}_t') = \nabla\ell_t(\mathbf{W}_t') \cdot (\mathbf{W}^\star - \mathbf{W}_t') + \frac{1}{2}(\mathbf{W}^\star - \mathbf{W}_t')^\top[\nabla^2\ell_t(\boldsymbol{\Psi})](\mathbf{W}^\star - \mathbf{W}_t'),$$

$$\geq ((\mathbf{p}_t - \mathbf{e}_{y_t}) \otimes \mathbf{x}_t) \cdot (\mathbf{W}^\star - \mathbf{W}_t') + \frac{1}{2}(\mathbf{W}^\star - \mathbf{W}_t')^\top[\frac{\alpha\delta}{10}\mathbf{H}_t \otimes \mathbf{x}_t\mathbf{x}_t^\top](\mathbf{W}^\star - \mathbf{W}_t'), \tag{12}$$

where the last inequality follows from (11). Finally, we have

$$\frac{1}{2}(\mathbf{W}^\star - \mathbf{W}_t')^\top[\eta\mathbf{H}_t \otimes \mathbf{x}_t\mathbf{x}_t^\top](\mathbf{W}^\star - \mathbf{W}_t') \leq \frac{1}{2}\eta\|\mathbf{H}_t \otimes \mathbf{x}_t\mathbf{x}_t^\top\|_2\|\mathbf{W}^\star - \mathbf{W}_t'\|^2 \leq 20\eta R^2 D^2, \tag{13}$$

since $\|\mathbf{W}^\star - \mathbf{W}_t'\| \leq 2D$. Adding inequalities (12) and (13), rearranging the result and using (7), (8), (9), and (10) gives the stated bound. $\qquad\square$

# 4  Experiments

While the theoretical regret bound for NEWTRON is $O(\log T)$ when $\alpha = O(1)$, the provable constant in $O(\cdot)$ notation is quite large, leading one to question the practical performance of the algorithm. The main reason for the large constant is that the analysis requires the $\beta$ parameter to be set extremely small to get the required bounds. In practice, however, one can keep $\beta$ a tunable parameter and try using larger values. In this section, we give experimental evidence (replicating the experiments of [KSST08]) that shows that the practical performance of the algorithm is quite good for small values of $\alpha$ (like 10), and not too small values of $\beta$ (like 0.01, 0.0001).

**Data sets.** We used three data sets from [KSST08]: SYNSEP, SYNNONSEP, and REUTERS4. The first two, SYNSEP and SYNNONSEP, are synthetic data sets, generated according to the description given in [KSST08]. These data sets have the same $10^6$ feature vectors with 400 features. There are 9 possible labels. The data set SYNSEP is linearly separable, whereas the data set SYNNONSEP is made inseparable by artificially adding 5% label noise. The REUTERS4 data set is generated from the Reuters RCV1 corpus. There are $673,768$ documents in the data set with 4 possible labels, and $346,810$ features. Our results are reported by averaging over 10 runs of the algorithm involved.

**Algorithms.** We implemented the BANDITRON and NEWTRON algorithms[3]. The NEWTRON algorithm is significantly slower than BANDITRON due to its quadratic running time. This makes it infeasible for really large data sets like REUTERS4. To surmount this problem, we implemented an approximate version of NEWTRON, called PNEWTRON[4], which runs in linear time per iteration and thus has comparable speed to BANDITRON. PNEWTRON does not have the same regret guarantees of NEWTRON however. To derive PNEWTRON, we can restate NEWTRON equivalently as (see [HAK07]):

$$\mathbf{W}_t' = \arg\min_{\mathbf{W} \in \mathcal{K}}(\mathbf{W} - \mathbf{W}_t'')^\top \mathbf{A}_t(\mathbf{W} - \mathbf{W}_t'')$$

where $\mathbf{W}_t'' = -\mathbf{A}_t^{-1}\mathbf{b}_t$, for $\mathbf{A}_t = \frac{1}{D}\mathbf{I} + \sum_{\tau=1}^{t-1}\kappa_\tau\beta\tilde{\boldsymbol{\nabla}}_\tau\tilde{\boldsymbol{\nabla}}_\tau^\top$ and $\mathbf{b}_t = \sum_{\tau=1}^{t-1}(1 - \kappa_\tau\beta\tilde{\boldsymbol{\nabla}}_\tau \cdot \mathbf{W}_\tau)\tilde{\boldsymbol{\nabla}}_\tau$. PNEWTRON makes the following change, using the diagonal approximation for the Hessian, and usual Euclidean projections:

$$\mathbf{W}_t' = \arg\min_{\mathbf{W} \in \mathcal{K}}(\mathbf{W} - \mathbf{W}_t'')^\top(\mathbf{W} - \mathbf{W}_t'')$$

where $\mathbf{W}_t'' = -\mathbf{A}_t^{-1}\mathbf{b}_t$, for $\mathbf{A}_t = \frac{1}{D}\mathbf{I} + \sum_{\tau=1}^{t-1} \mathrm{diag}(\kappa_\tau \beta \tilde{\boldsymbol{\nabla}}_\tau \tilde{\boldsymbol{\nabla}}_\tau^\top)$ and $\mathbf{b}_t$ is the same as before, $\mathbf{b}_t = \sum_{\tau=1}^{t-1}(1 - \kappa_\tau \beta \tilde{\boldsymbol{\nabla}}_\tau \cdot \mathbf{W}_\tau)\tilde{\boldsymbol{\nabla}}_\tau$.

**Parameter settings.** In our experiments, we chose $\mathcal{K}$ to be the unit $\ell_2$ ball in $\mathbb{R}^{kn}$, so $D = 1$. We also choose $\alpha = 10$ for all experiments in the log-loss. For BANDITRON, we chose the value of $\gamma$ specified in [KSST08]: $\gamma = 0.014, 0.006$ and $0.05$ for SYNSEP, SYNNONSEP and REUTERS4 respectively. For NEWTRON and PNEWTRON, we chose $\gamma = 0.01, 0.006$ and $0.05$ respectively. The other parameter for NEWTRON and PNEWTRON, $\beta$, was set to the values $\beta = 0.01, 0.01$, and $0.0001$ respectively. We did not tune any of the parameters $\alpha$, $\beta$ and $\gamma$ for NEWTRON or PNEWTRON.

**Evaluation.** We evaluated the algorithms in terms of their error rate, i.e. the fraction of prediction mistakes made as a function of time. Experimentally, PNEWTRON has quite similar performance to NEWTRON, but is significantly faster. Figure 4 shows how BANDITRON, NEWTRON and PNEWTRON compare on the SYNNONSEP data set for $10^4$ examples[5]. It can be seen that PNEWTRON has similar behavior to NEWTRON, and is not much worse.

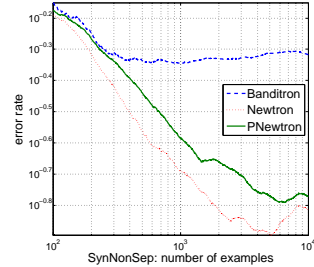

Figure 1: Log-log plots of error rates vs. number of examples for BANDITRON, NEWTRON and PNEWTRON on SYNNON-SEP with $10^4$ examples.

The rest of the experiments were conducted using only BAN-DITRON and PNEWTRON. The results are shown in figure 4. It can be clearly seen that PNEWTRON decreases the error rate much faster than BANDITRON. For the SYNSEP data set, PNEWTRON very rapidly converges to the lowest possible error rate due to setting the exploration parameter $\gamma = 0.01$, viz. $0.01 \times 8/9 = 0.89\%$. In comparison, the final error for BANDITRON is $1.91\%$. For the SYNNONSEP data set, PNEWTRON converges rapidly to its final value of $11.94\%$. BANDITRON remains at a high error level until about $10^4$ examples, and at the very end catches up with and does slightly better than PNEWTRON, ending at $11.47\%$. For the REUTERS4 data set, both BANDITRON and PNEWTRON decrease the error rate at roughly same pace; however PNEWTRON still obtains better performance consistently by a few percentage points. In our experiments, the final error rate for PNEWTRON is $13.08\%$, while that for BANDITRON is $18.10\%$.

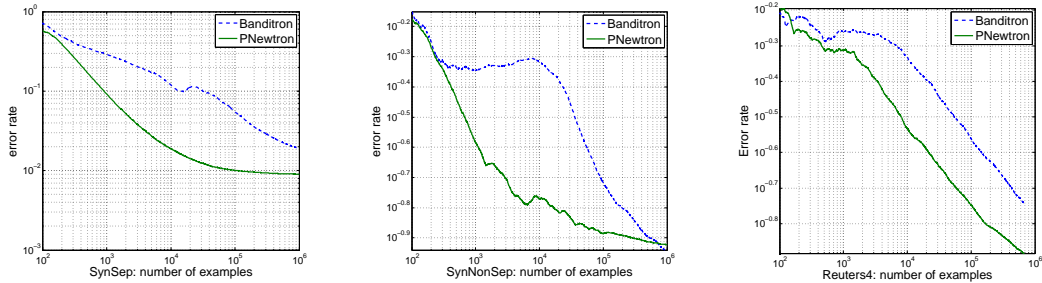

Figure 2: Log-log plots of error rates vs. number of examples for BANDITRON and PNEWTRON on different data sets. Left: SYNSEP. Middle: SYNNONSEP. Right: REUTERS4.

# 5  Future Work

Some interesting questions remain open. Our theoretical guarantee applies only to the quadratic-time NEWTRON algorithm. Is it possible to obtain similar regret guarantees for a linear time algorithm? Our regret bound has an exponentially large constant, which depends on the loss functions parameters. Does there exist an algorithm with similar regret guarantees but better constants?

## Footnotes

[1]For the basic bandit classification problem see [DHK07, RTB07, DH06, FKM05, AK08, MB04, AHR08].

[2] Our earlier proof used Cheeger's inequality. We thank an anonymous referee for a simplified proof.

[3]We did not implement the Confidit algorithm of [CG11] since our aim was to consider algorithms in the fully adversarial setting.

[4]Short for pseudo-NEWTRON. The "P" may be left silent so that it's almost NEWTRON, but not quite.

[5]In the interest of reducing running time for NEWTRON, we used a smaller data set.

# References

[ACBFS03] Peter Auer, Nicolò Cesa-Bianchi, Yoav Freund, and Robert E. Schapire. The non-stochastic multiarmed bandit problem. *SIAM J. Comput.*, 32:48–77, January 2003.

[AHR08] Jacob Abernethy, Elad Hazan, and Alexander Rakhlin. Competing in the dark: An efficient algorithm for bandit linear optimization. In *COLT*, pages 263–274, 2008.

[AK08] Baruch Awerbuch and Robert Kleinberg. Online linear optimization and adaptive routing. *J. Comput. Syst. Sci.*, 74(1):97–114, 2008.

[AR09] Jacob Abernethy and Alexander Rakhlin. An efficient bandit algorithm for $\sqrt{T}$-regret in online multiclass prediction? In *COLT*, 2009.

[CG11] Koby Crammer and Claudio Gentile. Multiclass classification with bandit feedback using adaptive regularization. In *ICML*, 2011.

[DH06] Varsha Dani and Thomas P. Hayes. Robbing the bandit: less regret in online geometric optimization against an adaptive adversary. In *SODA*, pages 937–943, 2006.

[DHK07] Varsha Dani, Thomas Hayes, and Sham Kakade. The price of bandit information for online optimization. In *NIPS*. 2007.

[FKM05] Abraham D. Flaxman, Adam Tauman Kalai, and H. Brendan McMahan. Online convex optimization in the bandit setting: gradient descent without a gradient. In *SODA*, pages 385–394, 2005.

[HAK07] Elad Hazan, Amit Agarwal, and Satyen Kale. Logarithmic regret algorithms for online convex optimization. *Machine Learning*, 69(2-3):169–192, 2007.

[HJ91] R.A. Horn and C.R. Johnson. *Topics in Matrix Analysis*. Cambridge University Press, Cambridge, 1991.

[KSST08] Sham M. Kakade, Shai Shalev-Shwartz, and Ambuj Tewari. Efficient bandit algorithms for online multiclass prediction. In *ICML'08*, pages 440–447, 2008.

[LZ07] John Langford and Tong Zhang. The epoch-greedy algorithm for multi-armed bandits with side information. In *NIPS*, 2007.

[MB04] H. Brendan McMahan and Avrim Blum. Online geometric optimization in the bandit setting against an adaptive adversary. In *COLT*, pages 109–123, 2004.

[RTB07] Alexander Rakhlin, Ambuj Tewari, and Peter Bartlett. Closing the gap between bandit and full-information online optimization: High-probability regret bound. Technical Report UCB/EECS-2007-109, EECS Department, University of California, Berkeley, Aug 2007.

